# Iterative Non-linear Dimensionality Reduction by Manifold Sculpting

**Mike Gashler, Dan Ventura, and Tony Martinez** *
Brigham Young University
Provo, UT 84604

## Abstract

Many algorithms have been recently developed for reducing dimensionality by projecting data onto an intrinsic non-linear manifold. Unfortunately, existing algorithms often lose significant precision in this transformation. Manifold Sculpting is a new algorithm that iteratively reduces dimensionality by simulating surface tension in local neighborhoods. We present several experiments that show Manifold Sculpting yields more accurate results than existing algorithms with both generated and natural data-sets. Manifold Sculpting is also able to benefit from both prior dimensionality reduction efforts.

## 1 Introduction

Dimensionality reduction is a two-step process: 1) Transform the data so that more information will survive the projection, and 2) project the data into fewer dimensions. The more relationships between data points that the transformation step is required to preserve, the less flexibility it will have to position the points in a manner that will cause information to survive the projection step. Due to this inverse relationship, dimensionality reduction algorithms must seek a balance that preserves information in the transformation without losing it in the projection. The key to finding the right balance is to identify where the majority of the information lies.

Nonlinear dimensionality reduction (NLDR) algorithms seek this balance by assuming that the relationships between neighboring points contain more informational content than the relationships between distant points. Although non-linear transformations have more potential than do linear transformations to lose information in the structure of the data, they also have more potential to position the data to cause more information to survive the projection. In this process, NLDR algorithms expose patterns and structures of lower dimensionality (manifolds) that exist in the original data. NLDR algorithms, or manifold learning algorithms, have potential to make the high-level concepts embedded in multidimensional data accessible to both humans and machines.

This paper introduces a new algorithm for manifold learning called *Manifold Sculpting*, which discovers manifolds through a process of progressive refinement. Experiments show that it yields more accurate results than other algorithms in many cases. Additionally, it can be used as a post-processing step to enhance the transformation of other manifold learning algorithms.

## 2 Related Work

Many algorithms have been developed for performing non-linear dimensionality reduction. Recent works include Isomap [1], which solves for an isometric embedding of data into fewer dimensions with an algebraic technique. Unfortunately, it is somewhat computationally expensive as it requires solving for the eigenvectors of a large dense matrix, and has difficulty with poorly sampled areas of

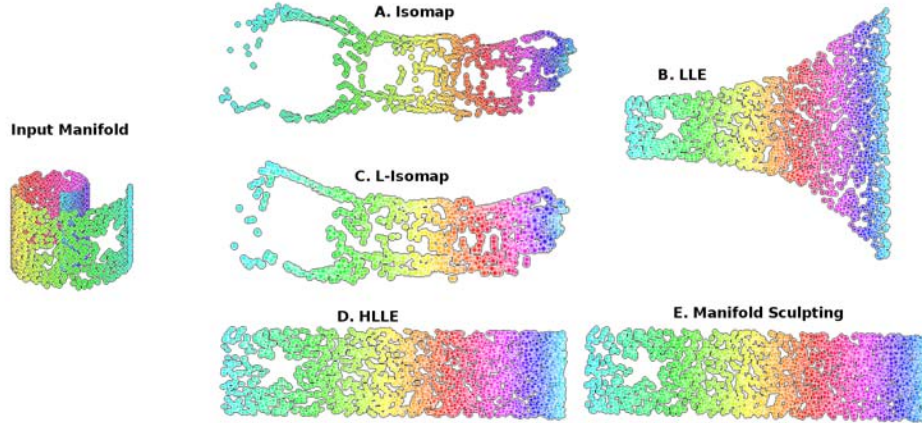

Figure 1: Comparison of several manifold learners on a Swiss Roll manifold. Color is used to indicate how points in the results correspond to points on the manifold. Isomap and L-Isomap have trouble with sampling holes. LLE has trouble with changes in sample density.

the manifold. (See Figure 1.A.) Locally Linear Embedding (LLE) [2] is able to perform a similar computation using a sparse matrix by using a metric that measures only relationships between vectors in local neighborhoods. Unfortunately it produces distorted results when the sample density is non-uniform. (See Figure 1.B.) An improvement to the Isomap algorithm was later proposed that uses landmarks to reduce the amount of necessary computation [3]. (See Figure 1.C.) Many other NLDR algorithms have been proposed, including Kernel Principle Component Analysis [4], Laplacian Eigenmaps [5], Manifold Charting [6], Manifold Parzen Windows [7], Hessian LLE [8], and others [9, 10, 11]. Hessian LLE preserves the manifold structure better than the other algorithms but is, unfortunately, computationally expensive. (See Figure 1.D.).

In contrast with these algorithms, *Manifold Sculpting* is robust to sampling issues and still produces very accurate results. This algorithm iteratively transforms data by balancing two opposing heuristics, one that scales information out of unwanted dimensions, and one that preserves local structure in the data. Experimental results show that this technique preserves information into fewer dimensions with more accuracy than existing manifold learning algorithms. (See Figure 1.E.)

# 3   The Algorithm

An overview of the Manifold Sculpting algorithm is given in Figure 2a.

| | |
|---|---|
| 1 | Find the $k$ nearest neighbors of each point. |
| 2 | Compute a set of relationships between neighbors. |
| 3 | Optionally align axes with principle components. |
| 4 | While the stopping criteria has not been met... |
| | a. Scale the data in the non-preserved dimensions. |
| | b. Adjust points to restore the relationships. |
| 5 | Project the data. |

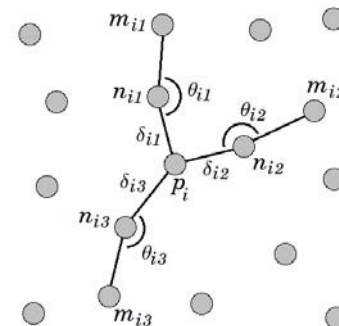

(a) A high-level overview of Manifold Sculpting.          (b) Local relationships

Figure 2: $\delta$ and $\theta$ define the relationships that Manifold Sculpting attempts to preserve.

**Step 1: Find the $k$ nearest neighbors of each point.** For each data point $p_i$ in $P$ (where $P$ is the set of all data points represented as vectors in $\mathbb{R}^n$), find the $k$-nearest neighbors $N_i$ (such that $n_{ij} \in N_i$ is the $j^{th}$ neighbor of point $p_i$).

**Step 2: Compute relationships between neighbors.** For each $j$ (where $0 < j \le k$) compute the Euclidean distance $\delta_{ij}$ between $p_i$ and each $n_{ij} \in N_i$. Also compute the angle $\theta_{ij}$ formed by the two line segments ($p_i$ to $n_{ij}$) and ($n_{ij}$ to $m_{ij}$), where $m_{ij}$ is the most colinear neighbor of $n_{ij}$ with $p_i$. (See Figure 2b.) The most colinear neighbor is the neighbor point that forms the angle closest to $\pi$. The values of $\delta$ and $\theta$ are the relationships that the algorithm will attempt to preserve during transformation. The global average distance between all the neighbors of all points $\delta_{ave}$ is also computed.

**Step 3: Optionally preprocess the data.** The data may optionally be preprocessed with the transformation step of Principle Component Analysis (PCA), or another efficient algorithm. Manifold Sculpting will work without this step; however, preprocessing can result in significantly faster convergence. To the extent that there is a linear component in the manifold, PCA will move the information in the data into as few dimensions as possible, thus leaving less work to be done in step 4 (which handles the non-linear component). This step is performed by computing the first $|D_{pres}|$ principle components of the data (where $D_{pres}$ is the set of dimensions that will be preserved in the projection), and rotating the dimensional axes to align with these principle components. (An efficient algorithm for computing principle components is presented in [12].)

**Step 4: Transform the data.** The data is iteratively transformed until some stopping criterion has been met. One effective technique is to stop when the sum change of all points during the current iteration falls below a threshold. The best stopping criteria depend on the desired quality of results – if precision is important, the algorithm may iterate longer; if speed is important it may stop earlier.

**Step 4a: Scale values.** All the values in $D_{scal}$ (The set of dimensions that will be eliminated by the projection) are scaled by a constant factor $\sigma$, where $0 < \sigma < 1$ ($\sigma = 0.99$ was used in this paper). Over time, the values in $D_{scal}$ will converge to 0. When $D_{scal}$ is dropped by the projection (step 5), there will be very little informational content left in these dimensions.

**Step 4b: Restore original relationships.** For each $p_i \in P$, the values in $D_{pres}$ are adjusted to recover the relationships that are distorted by scaling. Intuitively, this step simulates tension on the manifold surface. A heuristic error value is used to evaluate the current relationships among data points relative to the original relationships:

$$\epsilon_{p_i} = \sum_{j=0}^{k} w_{ij} \left( \left( \frac{\delta_{ij} - \delta_{ij_0}}{2\delta_{ave}} \right)^2 + \left( \frac{\theta_{ij} - \theta_{ij_0}}{\pi} \right)^2 \right) \tag{1}$$

where $\delta_{ij}$ is the current distance to $n_{ij}$, $\delta_{ij_0}$ is the original distance to $n_{ij}$ measured in step 2, $\theta_{ij}$ is the current angle, and $\theta_{ij_0}$ is the original angle measured in step 2. The denominator values were chosen as normalizing factors because the value of the angle term can range from 0 to $\pi$, and the value of the distance term will tend to have a mean of about $\delta_{ave}$ with some variance in both directions. We adjust the values in $D_{pres}$ for each point to minimize this heuristic error value.

The order in which points are adjusted has some impact on the rate of convergence. Best results were obtained by employing a breadth-first neighborhood graph traversal from a randomly selected point. (A new starting point is randomly selected for each iteration.) Intuitively this may be analogous to the manner in which a person smoothes a crumpled piece of paper by starting at an arbitrary point and smoothing outward. To further speed convergence, higher weight, $w_{ij}$, is given to the component of the error contributed by neighbors that have already been adjusted in the current iteration. For all of our experiments, we use $w_{ij} = 1$ if $n_i$ has not yet been adjusted in this iteration, and $w_{ij} = 10$, if $n_{ij}$ has been adjusted in this iteration.

Unfortunately the equation for the true gradient of the error surface defined by this heuristic is complex, and is in $O(|D|^3)$. We therefore use the simple hill-climbing technique of adjusting in each dimension in the direction that yields improvement.

Since the error surface is not necessarily convex, the algorithm may potentially converge to local minima. At least three factors, however, mitigate this risk: First, the PCA pre-processing step often tends to move the whole system to a state somewhat close to the global minimum. Even if a local

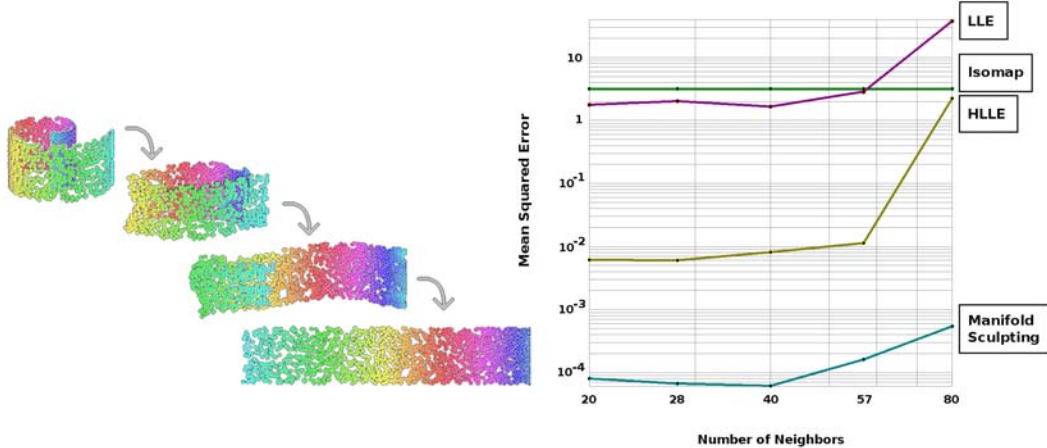

Figure 3: The mean squared error of four algorithms with a Swiss Roll manifold using a varying number of neighbors $k$. When $k > 57$, neighbor paths cut across the manifold. Isomap is more robust to this problem than other algorithms, but HLLE and Manifold Sculpting still yield better results. Results are shown on a logarithmic scale.

minimum exists so close to the globally optimal state, it may have a sufficiently small error as to be acceptable. Second, every point has a unique error surface. Even if one point becomes temporarily stuck in a local minimum, its neighbors are likely to pull it out, or change the topology of its error surface when their values are adjusted. Very particular conditions are necessary for every point to simultaneously find a local minimum. Third, by gradually scaling the values in $D_{scaled}$ (instead of directly setting them to 0), the system always remains in a state very close to the current globally optimal state. As long as it stays close to the current optimal state, it is unlikely for the error surface to change in a manner that permanently separates it from being able to reach the globally optimal state. (This is why all the dimensions need to be preserved in the PCA pre-processing step.) And perhaps most significantly, our experiments show that Manifold Sculpting generally tends to converge to very good results.

**Step 5: Project the data.** At this point $D_{scal}$ contains only values that are very close to zero. The data is projected by simply dropping these dimensions from the representation.

## 4 Empirical Results

Figure 1 shows that Manifold Sculpting appears visually to produce results of higher quality than LLE and Isomap with the Swiss Roll manifold, a common visual test for manifold learning algorithms. Quantitative analysis shows that it also yields better results than HLLE. Since the actual structure of this manifold is known prior to using any manifold learner, we can use this prior information to quantitatively measure the accuracy of each algorithm.

### 4.1 Varying number of neighbors.

We define a Swiss Roll in 3D space with $n$ points $(x_i, y_i, z_i)$ for each $0 \leq i < n$, such that $x_i = t \sin(t)$, $y_i$ is a random number $-6 \leq y_i < 6$, and $z_i = t \cos(t)$, where $t = 8i/n + 2$. In 2D manifold coordinates, the point is $(u_i, v_i)$, such that $u_i = \frac{\sinh^{-1}(t) + t\sqrt{t^2 + 1}}{2}$ and $v_i = y_i$.

We created a Swiss Roll with 2000 data points and reduced the dimensionality to 2 with each of four algorithms. Next we tested how well these results align with the expected values by measuring the mean squared distance from each point to its expected value. (See Figure 3.) We rotated, scaled, and translated the values as required to obtain the minimum possible error measurement for each algorithm. These results are consistent with a qualitative assessment of Figure 1. Results are shown with a varying number of neighbors $k$. In this example, when $k = 57$, local neighborhoods begin to cut across the manifold. Isomap is more robust to this problem than other algorithms, but HLLE and Manifold Sculpting still yield better results.

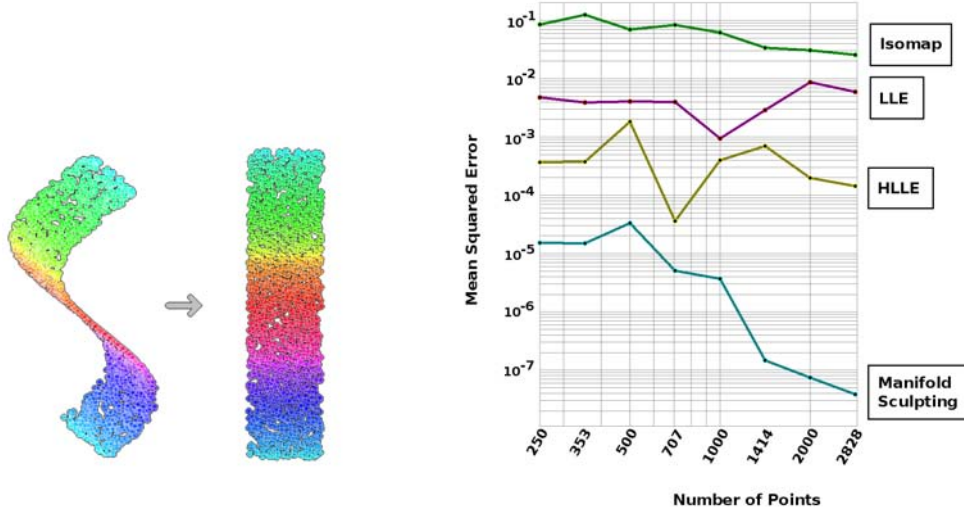

Figure 4: The mean squared error of points from an S-Curve manifold for four algorithms with a varying number of data points. Manifold Sculpting shows a trend of increasing accuracy with an increasing number of points. This experiment was performed with 20 neighbors. Results are shown on a logarithmic scale.

## 4.2 Varying sample densities.

A similar experiment was performed with an S-Curve manifold. We defined the S-Curve points in 3D space with $n$ points $(x_i, y_i, z_i)$ for each $0 \leq i < n$, such that $x_i = t$, $y_i = sin(t)$, and $z_i$ is a random number $0 \leq z_i < 2$, where $t = \frac{(2.2i - 0.1)\pi}{n}$. In 2D manifold coordinates, the point is $(u_i, v_i)$, such that $u_i = \int_0^t \left( \sqrt{\cos^2(w) + 1} \right) dw$ and $v_i = y_i$.

Figure 4 shows the mean squared error of the transformed points from their expected values using the same regression technique described for the experiment with the Swiss Roll problem. We varied the sampling density to show how this affects each algorithm. A trend can be observed in this data that as the number of sample points increases, the quality of results from Manifold Sculpting also increases. This trend does not appear in the results from other algorithms.

One drawback to the Manifold Sculpting algorithm is that convergence may take longer when the value for $k$ is too small. This experiment was also performed with 6 neighbors, but Manifold Sculpting did not always converge within a reasonable time when so few neighbors were used. The other three algorithms do not have this limitation, but the quality of their results still tend to be poor when very few neighbors are used.

## 4.3 Entwined spirals manifold.

A test was also performed with an Entwined Spirals manifold. In this case, Isomap was able to produce better results than Manifold Sculpting (see Figure 5), even though Isomap yielded the worst accuracy in previous problems. This can be attributed to the nature of the Isomap algorithm. In cases where the manifold has an intrinsic dimensionality of exactly 1, a path from neighbor to neighbor provides an accurate estimate of isolinear distance. Thus an algorithm that seeks to globally optimize isolinear distances will be less susceptible to the noise from cutting across local corners. When the intrinsic dimensionality is higher than 1, however, paths that follow from neighbor to neighbor produce a zig-zag pattern that introduces excessive noise into the isolinear distance measurement. In these cases, preserving local neighborhood relationships with precision yields better overall results than globally optimizing an error-prone metric. Consistent with this intuition, Isomap is the closest competitor to Manifold Sculpting in other experiments that involved a manifold with a single intrinsic dimension, and yields the poorest results of the four algorithms when the intrinsic dimensionality is larger than one.

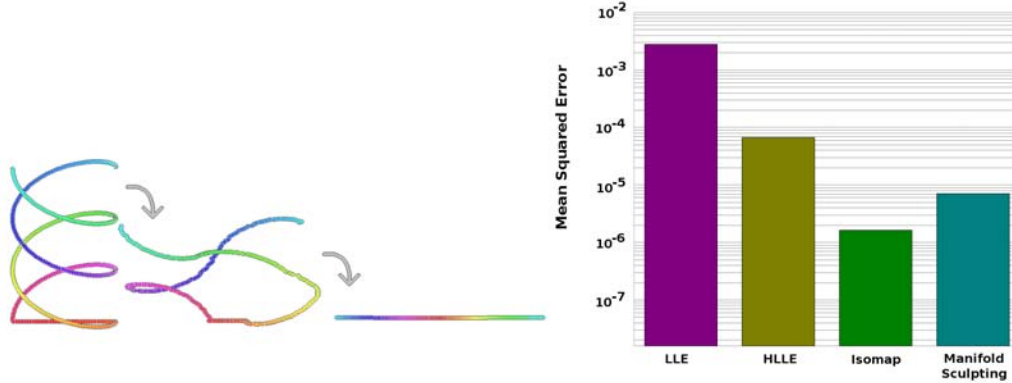

Figure 5: Mean squared error for four algorithms with an Entwined Spirals manifold.

## 4.4 Image-based manifolds.

The accuracy of Manifold Sculpting is not limited to generated manifolds in three dimensional space. Unfortunately, the manifold structure represented by most real-world problems is not known *a priori*. The accuracy of a manifold learner, however, can still be estimated when the problem involves a video sequence by simply counting the percentage of frames that are sorted into the same order as the video sequence. Figure 6 shows several frames from a video sequence of a person turning his head while gradually smiling. Each image was encoded as a vector of $1,634$ pixel intensity values. This data was then reduced to a single dimension. (Results are shown on three separate lines in order to fit the page.) The one preserved dimension could then characterize each frame according to the high-level concepts that were previously encoded in many dimensions. The dot below each image corresponds to the single-dimensional value in the preserved dimension for that image. In this case, the ordering of every frame was consistent with the video sequence.

## 4.5 Controlled manifold topologies.

Figure 7 shows a comparison of results obtained from a manifold generated by translating an image over a background of random noise. Nine of the 400 input images are shown as a sample, and results with each algorithm are shown as a mesh. Each vertex is placed at a position corresponding to the two values obtained from one of the 400 images. For increased visibility of the inherent structure, the vertexes are connected with their nearest input space neighbors. Because two variables (horizontal position and vertical position) were used to generate the dataset, this data creates a manifold with an intrinsic dimensionality of two in a space with an extrinsic dimensionality of 2,401 (the total number of pixels in each image). Because the background is random, the average distance between neighboring points in the input space is uniform, so the ideal result is known to be a square. The distortions produced by Manifold Sculpting tend to be local in nature, while the distortions produced by other algorithms tend to be more global. Note that the points are spread nearly uniformly across the manifold in the results from Manifold Sculpting. This explains why the results from Manifold Sculpting tend to fit the ideal results with much lower total error (as shown in

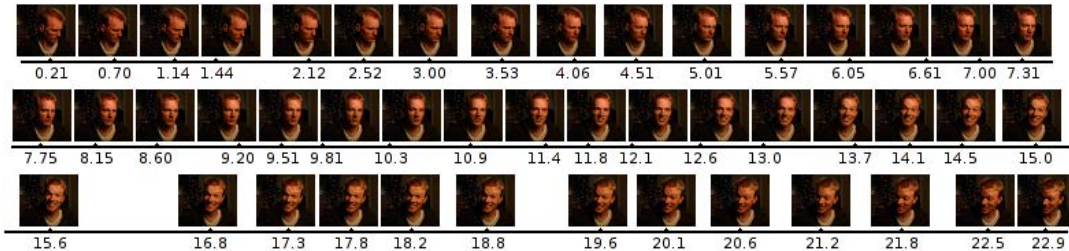

Figure 6: Images of a face reduced by Manifold Sculpting into a single dimension. The values are are shown here on three wrapped lines in order to fit the page. The original image is shown above each point.

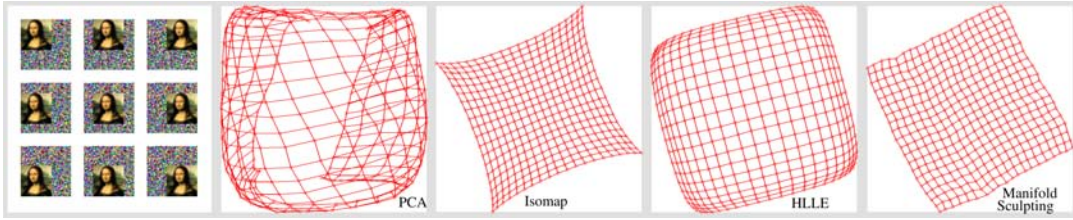

Figure 7: A comparison of results with a manifold generated by translating an image over a background of noise. Manifold Sculpting tends to produce less global distortion, while other algorithms tend to produce less local distortion. Each point represents an image. This experiment was done in each case with 8 neighbors. (LLE fails to yield results with these parameters, but [13] reports a similar experiment in which LLE produces results. In that case, as with Isomap and HLLE as shown here, distortion is clearly visible near the edges.)

Figure 3 and Figure 4). Perhaps more significantly, it also tends to keep the intrinsic variables in the dataset more linearly separable. This is particularly important when the dimensionality reduction is used as a pre-processing step for a supervised learning algorithm.

We created four video sequences designed to show various types of manifold topologies and measured the accuracy of each manifold learning algorithm. These results (and sample frames from each video) are shown in Figure 8. The first video shows a rotating stuffed animal. Since the background pixels remain nearly constant while the pixels on the rotating object change in value, the manifold corresponding to the vector encoding of this video will contain both smooth and changing areas. The second video was made by moving a camera down a hallway. This produces a manifold with a continuous range of variability, since pixels near the center of the frame change slowly while pixels near the edges change rapidly. The third video pans across a scene. Unlike the video of the rotating stuffed animal, there are no background pixels that remain constant. The last video shows another rotating stuffed animal. Unlike the first video, however, the high-contrast texture of the object used in this video results in a topology with much more variation. As the black spots shift across the pixels, a manifold is created that swings wildly in the respective dimensions. Due to the large hills and valleys in the topology of this manifold, the nearest neighbors of a frame frequently create paths that cut across the manifold. In all four cases, Manifold Sculpting produced results competitive with Isomap, which does particularly well with manifolds that have an intrinsic dimensionality of

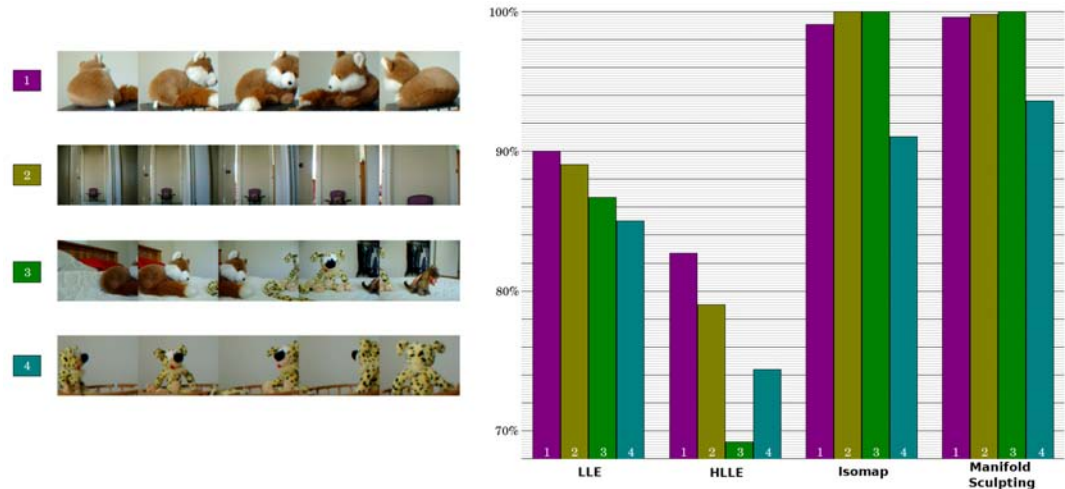

Figure 8: Four video sequences were created with varying properties in the corresponding manifolds. Dimensionality was reduced to one with each of four manifold learning algorithms. The percentage of frames that were correctly ordered by each algorithm is shown.

one, but Manifold Sculpting is not limited by the intrinsic dimensionality as shown in the previous experiments.

## 5 Discussion

The experiments tested in this paper show that Manifold Sculpting yields more accurate results than other well-known manifold learning algorithms. Manifold Sculpting is robust to holes in the sampled area. Manifold Sculpting is more accurate than other algorithms when the manifold is sparsely sampled, and the gap is even wider with higher sampling densities. Manifold Sculpting has difficulty when the selected number of neighbors is too small but consistently outperforms other algorithms when it is larger.

Due to the iterative nature of Manifold Sculpting, it's difficult to produce a valid complexity analysis. Consequently, we measured the scalability of Manifold Sculpting empirically and compared it with that of HLLE, L-Isomap, and LLE. Due to space constraints these results are not included here, but they indicate that Manifold Sculpting scales better than the other algorithms when when the number of data points is much larger than the number of input dimensions.

Manifold Sculpting benefits significantly when the data is pre-processed with the transformation step of PCA. The transformation step of any algorithm may be used in place of this step. Current research seeks to identify which algorithms work best with Manifold Sculpting to efficiently produce high quality results. (An implementation of Manifold Sculpting is included at *http://waffles.sourceforge.net*.)

## Footnotes

*mikegashler@gmail.com, ventura@cs.byu.edu, martinez@cs.byu.edu

## References

[1] Joshua B. Tenenbaum, Vin de Silva, and John C. Langford. A global geometric framework for nonlinear dimensionality reduction. *Science*, 290:2319–2323, 2000.

[2] Sam T. Roweis and Lawrence K. Saul. Nonlinear dimensionality reduction by locally linear embedding. *Science*, 290:2323–2326, 2000.

[3] Vin de Silva and Joshua B. Tenenbaum. Global versus local methods in nonlinear dimensionality reduction. In *NIPS*, pages 705–712, 2002.

[4] Bernhard Schölkopf, Alexander J. Smola, and Klaus-Robert Müller. Kernel principal component analysis. *Advances in kernel methods: support vector learning*, pages 327–352, 1999.

[5] Mikhail Belkin and Partha Niyogi. Laplacian eigenmaps and spectral techniques for embedding and clustering. In *Advances in Neural Information Processing Systems, 14*, pages 585–591, 2001.

[6] Matthew Brand. Charting a manifold. In *Advances in Neural Information Processing Systems, 15*, pages 961–968. MIT Press, Cambridge, MA, 2003.

[7] Pascal Vincent and Yoshua Bengio. Manifold parzen windows. In *Advances in Neural Information Processing Systems 15*, pages 825–832. MIT Press, Cambridge, MA, 2003.

[8] D. Donoho and C. Grimes. Hessian eigenmaps: locally linear embedding techniques for high dimensional data. *Proc. of National Academy of Sciences*, 100(10):5591–5596, 2003.

[9] Yoshua Bengio and Martin Monperrus. Non-local manifold tangent learning. In *Advances in Neural Information Processing Systems 17*, pages 129–136. MIT Press, Cambridge, MA, 2005.

[10] Elizaveta Levina and Peter J. Bickel. Maximum likelihood estimation of intrinsic dimension. In *NIPS*, 2004.

[11] Zhenyue Zhang and Hongyuan Zha. A domain decomposition method for fast manifold learning. In Y. Weiss, B. Schölkopf, and J. Platt, editors, *Advances in Neural Information Processing Systems 18*. MIT Press, Cambridge, MA, 2006.

[12] Sam Roweis. Em algorithms for PCA and SPCA. In Michael I. Jordan, Michael J. Kearns, and Sara A. Solla, editors, *Advances in Neural Information Processing Systems*, volume 10, 1998.

[13] Lawrence K. Saul and Sam T. Roweis. Think globally, fit locally: Unsupervised learning of low dimensional manifolds. *Journal of Machine Learning Research*, 4:119–155, 2003.

